# Natural Images, Gaussian Mixtures and Dead Leaves

**Daniel Zoran**
Interdisciplinary Center for Neural Computation
Hebrew University of Jerusalem
Israel
http://www.cs.huji.ac.il/ daniez

**Yair Weiss**
School of Computer Science and Engineering
Hebrew University of Jerusalem
Israel
yweiss@cs.huji.ac.il

## Abstract

Simple Gaussian Mixture Models (GMMs) learned from pixels of natural image patches have been recently shown to be surprisingly strong performers in modeling the statistics of natural images. Here we provide an in depth analysis of this simple yet rich model. We show that such a GMM model is able to compete with even the most successful models of natural images in log likelihood scores, denoising performance and sample quality. We provide an analysis of what such a model learns from natural images as a function of number of mixture components — including covariance structure, contrast variation and intricate structures such as textures, boundaries and more. Finally, we show that the salient properties of the GMM learned from natural images can be derived from a simplified Dead Leaves model which explicitly models occlusion, explaining its surprising success relative to other models.

## 1 GMMs and natural image statistics models

Many models for the statistics of natural image patches have been suggested in recent years. Finding good models for natural images is important to many different research areas — computer vision, biological vision and neuroscience among others. Recently, there has been a growing interest in comparing different aspects of models for natural images such as log-likelihood and multi-information reduction performance, and much progress has been achieved [1, 2, 3, 4, 5, 6]. Out of these results there is one which is particularly interesting: simple, unconstrained Gaussian Mixture Models (GMMs) with a relatively small number of mixture components learned from image patches are extraordinarily good in modeling image statistics [6, 4]. This is a surprising result due to the simplicity of GMMs and their ubiquity. Another surprising aspect of this result is that many of the current models may be thought of as GMMs with an exponential or infinite number of components, having different constraints on the covariance structure of the mixture components.

In this work we study the nature of GMMs learned from natural image patches. We start with a thorough comparison to some popular and cutting edge image models. We show that indeed, GMMs are excellent performers in modeling natural image patches. We then analyze what properties of natural images these GMMs capture, their dependence on the number of components in the mixture and their relation to the structure of the world around us. Finally, we show that the learned GMM suggests a strong connection between natural image statistics and a simple variant of the dead leaves model [7, 8], explicitly modeling occlusions and explaining some of the success of GMMs in modeling natural images.

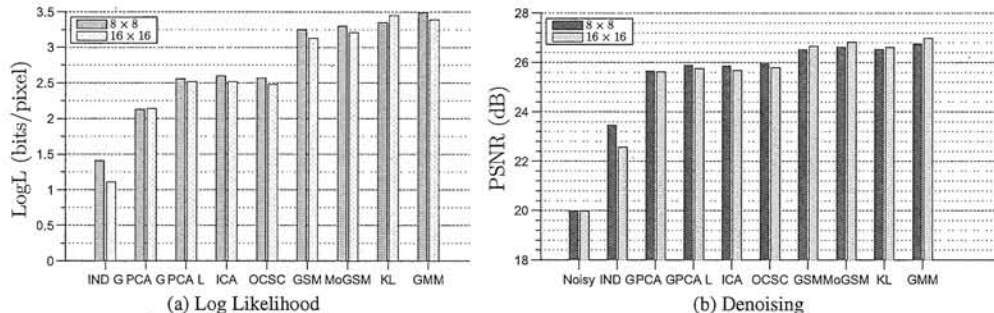

<div style="text-align:center">(a) Log Likelihood        (b) Denoising</div>

Figure 1: **(a)** Log likelihood comparison - note how the GMM is able to outperform (or equal) all other models despite its simplicity. **(b)** Denoising performance comparison - the GMM outperforms all other models here as well, and denoising performance is more or less consistent with likelihood performance. See text for more details.

## 2 Natural image statistics models - a comparison

As a motivation for this work, we start by rigorously comparing current models for natural images with GMMs. While some comparisons have been reported before with a limited number of components in the GMM [6], we want to compare to state-of-the-art models also varying the number of components systematically.

Each model was trained on $8 \times 8$ or $16 \times 16$ patches randomly sampled from the Berkeley Segmentation Database *training* images (a data set of millions of patches). The DC component of all patches was removed, and we discard it in all calculations. In all experiments, evaluation was done on the same, *unseen* test set of a 1000 patches sampled from the Berkeley *test* images. We removed patches having standard deviation below 0.002 (intensity values are between 0 and 1) as these are totally flat patches due to saturation and contain no structure (only 8 patches were removed from the test set). We do not perform any further preprocessing. The models we compare are: White Gaussian Noise (**Ind. G**), PCA/Gaussian (**PCA G**), PCA/Laplace (**PCA L**), ICA (**ICA**) [9, 10, 11], $2\times$Overcomplete sparse coding ($2\times$**OCSC**) [9], Gaussian Scale Mixture (**GSM**), Mixture of Gaussian Scale Mixture (**MoGSM**) [6], Karklin and Lewicki (**KL**) [12] and the **GMM** (with 200 components).

We compare the models using three criteria - log likelihood on unseen data, denoising results on unseen data and visual quality of samples from each model. The complete details of training, testing and comparisons may be found in the supplementary material of this paper - we encourage the reader to read these details. All models and code are available online at: www.cs.huji.ac.il/~daniez

**Log likelihood** The first experiment we conduct is a log likelihood comparison. For most of the models above, a closed form calculation of the likelihood is possible, but for the $2\times$OCSC and KL models, we resort to Hamiltonian Importance Sampling (HAIS) [13]. HAIS allows us to estimate likelihoods for these models accurately, and we have verified that the approximation given by HAIS is relatively accurate in cases where exact calculations are feasible (see supplementary material for details). The results of the experiment may be seen in Figure 1a. There are several interesting results in this figure. First, the important thing to note here is that GMMs outperforms all of the models and is similar in performance to Karklin and Lewicki. In [6] a GMM with far less components (2-5) has been compared to some other models (notably Restricted Boltzman Machines which the GMM outperforms, and MoGSMs which slightly outperform the GMMs in this work). Second, ICA with its learned Gabor like filters [10] gives a very minor improvement when compared to PCA filters with the same marginals. This has been noted before in [1]. Finally, overcomplete sparse coding is actually a bit *worse* than complete sparse coding - while this is counter intuitive, this result has been reported before as well [14, 2].

**Denoising** We compare the denoising performance of the different models. We added independent white Gaussian noise with known standard deviation $\sigma_n = 25/255$ to each of the patches in the test set **x**. We then calculate the MAP estimate $\hat{\mathbf{x}}$ of each model given the noisy patch. This can

be done in closed form for some of the models, and for those models where the MAP estimate does not have a closed form, we resort to numerical approximation (see supplementary material for more details). The performance of each model was measured using Peak Signal to Noise Ratio (PSNR): $\text{PSNR} = \log_{10}\left(\frac{1}{\|\mathbf{x}-\hat{\mathbf{x}}\|^2}\right)$ . Results can be seen in Figure 1b. Again, the GMM performs extraordinarily well, outperforming all other models. As can be seen, results are consistent with the log likelihood experiment - models with better likelihood tend to perform better in denoising [4].

**Sample Quality**   As opposed to log likelihood and denoising, generating samples from all the models compared here is easy. While it is more of a subjective measure, the visual quality of samples may be an indicator to how well interesting structures are captured by a model. Figure 2 depicts $16 \times 16$ samples from a subset of the models compared here. Note that the GMM samples capture a lot of the structure of natural images such as edges and textures, visible on the far right of the figure. The Karklin and Lewicki model produces rather structured patches as well. GSM seems to capture the contrast variation of images, but the patches themselves have very little structure (similar results obtained with MoGSM, not shown). PCA lacks any meaningful structure, other than $1/f$ power spectrum.

As can be seen in the results we have just presented, the GMM is a very strong performer in modeling natural image patches. While we are not claiming Gaussian Mixtures are the *best* models for natural images, we do think this is an interesting result, and as we shall see later, it relates intimately to the structure of natural images.

# 3   Analysis of results

So far we have seen that despite their simplicity, GMMs are very capable models for natural images. We now ask - what do these models learn about natural images, and how does this affect their performance?

## 3.1   How many mixture components do we need?

While we try to learn our GMMs with as few a priori assumptions as possible, we do need to set one important parameter - the number of components in the mixture. As noted above, many of the current models of natural images can be written in the form of GMMs with an exponential or infinite number of components and different kinds of constraints on the covariance structure. Given this, it is quite surprising that a GMM with a relatively small number of component (as above) is able to compete with these models. Here we again evaluate the GMM as in the previous section but now systematically vary the number of components and the size of the image patch. Results for the $16 \times 16$ model are shown in figure 3, see supplementary material for other patch sizes.

As can be seen, moving from one component to two already gives a tremendous boost in performance, already outperforming ICA but still not enough to outperform GSM, which is outperformed at around 16 components. As we add more and more components to the mixture performance increases, but seems to be converging to some upper bound (which is not reached here, see supplementary material for smaller patch sizes where it is reached). This shows that a small number of components is indeed

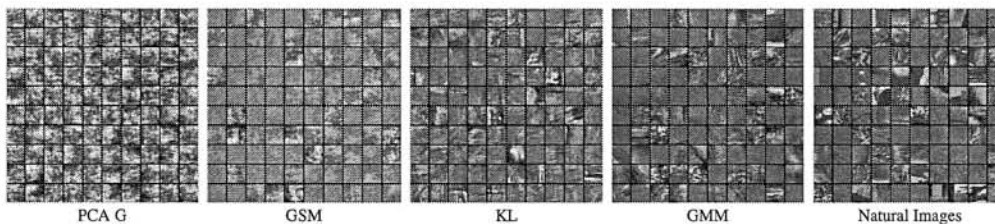

| PCA G | GSM | KL | GMM | Natural Images |

Figure 2: Samples generated from some of the models compared in this work. PCA G produces no structure other than $1/f$ power spectrum. GSM capture the contrast variation of image patches nicely, but the patches themselves have no structure. The GMM and KL models produce quite structured patches - compare with the natural image samples on the right.

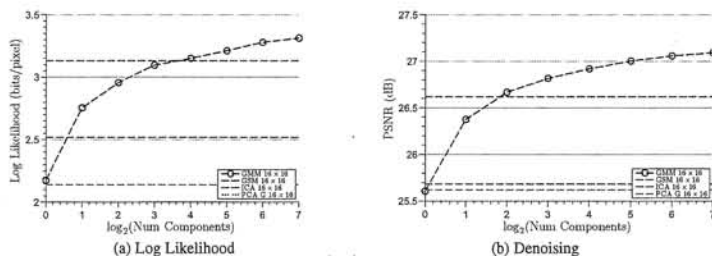

(a) Log Likelihood          (b) Denoising

Figure 3: **(a)** Log likelihood of GMMs trained on natural image patches, as a function of the number of components in the mixture. Models of $16 \times 16$ were trained on a training set. Likelihood was calculated on an unseen test set of patches. Already at 2 components the GMM outperforms ICA and at 16 components it outperforms the 16 component GSM model. Likelihood continues to improve as we add more components. See supplementary material for other patch sizes. **(b)** Denoising performance as a function of number of components - performance behave qualitatively the same as likelihood.

sufficient to achieve good performance and begs the questions - what do the first few components learn that gives this boost in performance? what happens when we add more components to the mixture, further improving performance? Before we answer these questions, we will shortly discuss what are the properties of GMMs which we need to examine to gain this understanding.

### 3.2 GMMs as generative models

In order to gain a better understanding of GMMs it will be useful to think of them from a generative perspective. The process of generating a sample from a GMM is a two step procedure; a non-linear one, and a linear one. We pick one of the mixture components - the chances for the $k$-th component to be picked are its mixing weight $\pi_k$. Having picked the $k$-th component, we now sample $N$ *independent* Gaussian variables with zero mean and unit variance, where $N$ is the number of pixels in a patch (minus one for the DC component). We arrange these coefficients into a vector $\mathbf{z}$. From the covariance matrix of the $k$-th component we calculate the eigenvector matrix $\mathbf{V}_k$ and eigenvalue matrix $\mathbf{D}_k$. Then, the new sample $\mathbf{x}$ is:

$$\mathbf{x} = \mathbf{V_k}\mathbf{D}_k^{0.5}\mathbf{z}$$

This tells us that we can think of each covariance matrix in the mixture as a dictionary with $N$ elements. The dictionary elements are the "directions" each eigenvector in patch space points to, and each of those is scaled by the corresponding eigenvalue. These are linearly mixed to form our patch. In other words, to gain a better understanding of what each mixture component is capturing, we need to look at the *eigenvectors* and *eigenvalues* of its corresponding covariance matrix.

### 3.3 Contrast

Figure 4 shows the eigenvectors and eigenvalues of the covariance matrices of a 2 component mixture - as can be seen, the eigenvectors of both mixture components are very similar and they differ only in their eigenvalue spectrum. The eigenvalue spectrum, on the other hand, is very similar in *shape* but differs by a multiplicative constant (note the log scale). This behavior remains the same as we add more and more components to the mixture — up to around 8-10 components (depending on the patch size, not shown here) we get more components with similar eigenvector structure but different eigenvalue distributions.

Modeling a patch as a mixture with the same eigenvectors but eigenvalues differing by a scalar multiplier is in fact equivalent to saying that each patch is the product of a scalar z and a multivariate Gaussian. This is exactly the Gaussian Scale Mixture model we compared to earlier! As can be seen, 8–10 components are already enough to equal the performance of the 16 component GSM. This means that what the first few components of the mixture capture is the contrast variability of natural image patches. This also means that factorial models like ICA have no hope in capturing this as contrast is a global scaling of all coefficients together (something which is highly unlikely under factorial models).

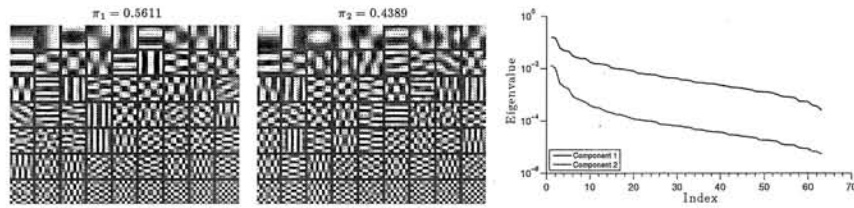

Figure 4: Eigenvectors and eigenvalues of covariance matrices in a 2 component GMM trained on natural images. Eigenvectors are sorted according to decreasing eigenvalue order, top left is the largest eigenvalue. Note that the two components have approximately the same eigenvectors (up to sign, and both resembling the Fourier basis) but different eigenvalue spectra. The eigenvalues mostly differ by a scalar multiplication (note the log scale), hinting that this is, in fact, approximately a GSM (see text for details).

## 3.4 Textures and boundaries

We have seen that the first components in the GMM capture the contrast variation of natural images, but as we saw in Figure 3, likelihood continues to improve as we add more components, so we ask: what do these extra components capture?

As we add more components to the mixture, we start revealing more specialized components which capture different properties of natural images. Sorting the components by their mixing weights (where the most likely ones are first), we observe that the first few tens of components are predominantly Fourier like components, similar to what we have seen thus far, with varying eigenvalue spectra. These capture *textures* at different scales and orientations. Figure 5 depicts two of these texture components - note how their eigenvector structure is similar, but samples sampled from each of them reveal that they capture different textures due to different eigenvalue spectra.

A more interesting family of components can be found in the mixture as we look into more rare components. These components model *boundaries* of objects or textures — their eigenvectors are structured such that most of the variability is on one side of an edge crossing the patch. These edges come at different orientations, shifts and contrasts. Figure 5 depicts some of these components at different orientations, along with two flat texture components for comparison. As can be seen, we obtain a Fourier like structure which is concentrated on one side of the patch. Sampling from the Gaussian associated with each mixture component (bottom row) reveals what each component actually captures - patches with different textures on each side of an edge.

To see how these components relate to actual structure in natural images we perform the following experiment. We take an unseen natural image, and for each patch in the image we calculate the most likely component from the learned mixture. Figure 6 depicts those patches assigned to each of the five components in Figure 5, where we show only non-overlapping patches for clarity (there are many more patches assigned to each component in the image). The colors correspond to each of the components in Figure 5. Note how the boundary components capture different orientations, and prefer mostly borders with a specific ordering (top to bottom edge, and not vice versa for example), while texture components tend to stay within object boundaries. The sources for these phenomena will be discussed in the next section.

# 4 The "mini" dead leaves model

## 4.1 Dead leaves models

We now show that many of the properties of natural scenes that were captured by the GMM model can be derived from a variant of the dead leaves model [15]. In the original dead leaves model, two dimensional textured surfaces (which are sometimes called "objects" or "leaves") are sampled from a shape and size distribution and then placed on the image plane at random positions, *occluding* one another to produce an image. With a good choice of parameters, such a model creates images which

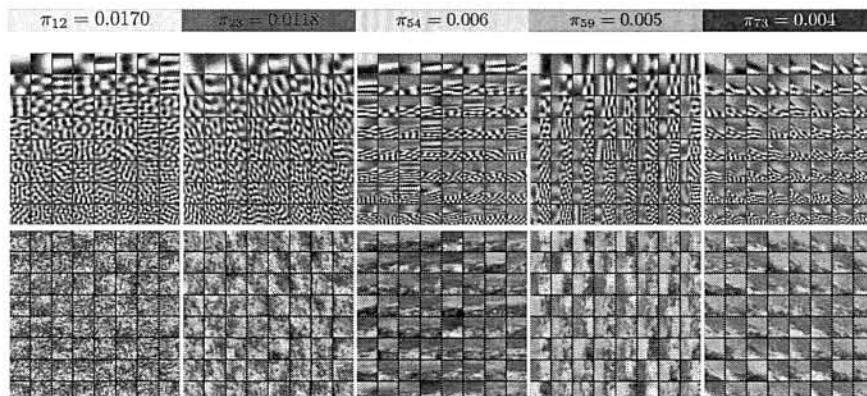

| $\pi_{12} = 0.0170$ | $\pi_{23} = 0.0118$ | $\pi_{54} = 0.006$ | $\pi_{59} = 0.005$ | $\pi_{73} = 0.004$ |

Figure 5: Leading eigenvectors (top row) and samples (bottom row) from 5 different components from a $16 \times 16$ GMM. From left to right: components 12 and 23, having a similar Fourier like eigenvector structure, but different eigenvalue spectra, notable by different texture generated from each component. Three different "boundary" like component: note how the eigenvector structure has a Fourier like structure which is concentrated only on side of the patch, depicting an edge structure. These come in different orientations, shifts and contrasts in the mixture. The color markings are in reference to Figure 6.

share many properties with natural images such as scale invariance, heavy tailed filter responses and bow-tie distributions for conditional pair-wise filter responses [16, 17, 8]. A recent work by Pitkow [8] provides an interesting review and analysis of these properties.

## 4.2 Mini dead leaves

We propose here a simple model derived from the dead leaves model which we call the "Mini Dead Leaves" model. This is a patch based version of the dead leaves model, and can be seen as an approximation of what happens when sampling small patches from an image produced by the dead leaves model.

In mini dead leaves we generate an image patch in the following manner: for each patch we randomly decide if this patch would be a "flat" patch or an "edge" patch. This is done by flipping a coin with probability $p$. **Flat** patches are then produced by sampling a texture from a given texture process. In this case we use a multidimensional Gaussian with some stationary texture covariance matrix which is multiplied by a scalar contrast variable. We then add to the texture a random scalar mean value, such that the final patch $\mathbf{x}$ is of the form: $\mathbf{x} = \mu + z\mathbf{t}$ where $\mu \sim \mathcal{N}(0,1)$ is a scalar, $\mathbf{t} \sim \mathcal{N}(\mathbf{0}, \Sigma)$ is a

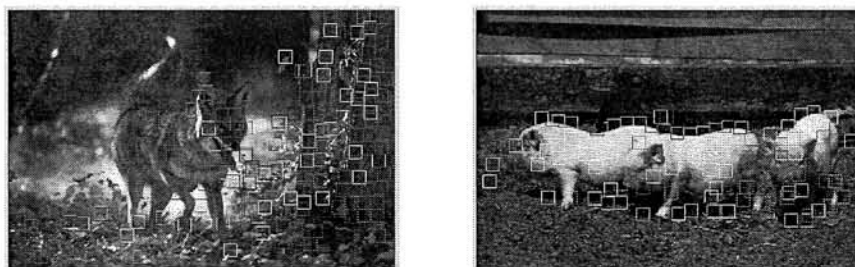

Figure 6: Components assignment on natural images taken from the Berkeley *test* images. For each patch in the image the most likely component from the mixture was calculated - presented here are patches which were assigned to one of the components in Figure 5. Assignment are much more dense than presented here, but we show only non-overlapping patches for clarity. Color codes correspond to the colors in Figure 5. Note how different components capture different structures in the image. See text and Figure 5 for more details.

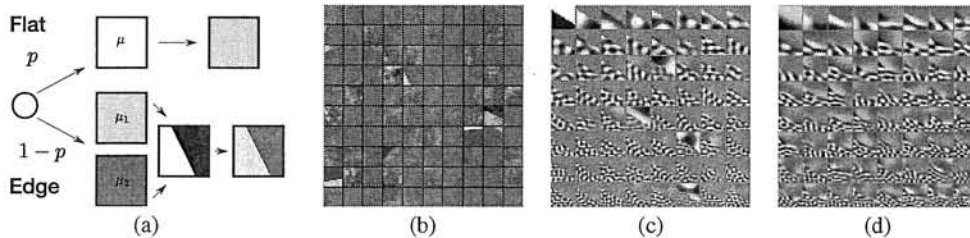

(a)  (b)  (c)  (d)

Figure 7: **(a)** The mini dead leaves models. Patches are either "flat" or "edge" patches. **Flat** patches are sampled from a multivariate Gaussian texture which is scaled by a contrast scalar and a mean value is added to it to form the patch. **Edge** patches are created by sampling two flat patches, an occlusion mask and setting the pixels of each side of the mask to come from a different flat patch. See text for full details. **(b)** Samples generated from the mini dead leaves model with their DC removed. **(c)** Leading eigenvectors of an edge component from a mini dead leaves model. **(d)** Leading eigenvectors of a component from the GMM trained on natural images - note how similar the structure is to the mini dead leaves model analytical result. See text for details.

vector and the scalar $z$ is sampled from a discrete set of variables $z_k$ with a corresponding probability $\pi_k$. This results in a GSM texture to which we add a random mean (DC) value. In all experiments here, we use a GSM trained on natural images.

**Edge** patches are generated by sampling two independent **Flat** patches from the texture process, $\mathbf{f}$ and $\mathbf{g}$, and then generating an *occlusion* mask to combine the two. We use a simple occlusion mask generation process here: we choose a random angle $\theta$ and a random distance $r$ measured from the center of the patch, where both $\theta$ and $r$ may be quantized — this defines the location of a straight edge on the patch. Every pixel on one side of the edge is assumed to come from the same object, and pixels from different sides of the patch come from different objects. We label all pixels belonging to one object by $L_1$ and to the other object by $L_2$. We then generate the patch by taking all pixels $i \in L_1$ to $x_i = f_i$ and similarly $x_{i \in L_2} = g_i$. This results in a patch with two textured areas, one with a mean value $\mu_1$ and the other with $\mu_2$. Figure 7a depicts the generative process for both kind of patches and Figure 7b depicts samples from the model.

### 4.3 Gaussian mixtures and dead leaves

It can be easily seen that the mini dead leaves model is, in fact, a GMM. For each configuration of hidden variables (denoting whether the patch is "flat" or "edge", the scalar multiplier $z$ and if it is an edge patch the second scalar multiplier $z_2$, $r$ and $\theta$) we have a Gaussian for which we *know* the covariance matrix *exactly*. Together, all configurations form a GMM - the interesting thing here is how the *structure* of the covariance matrix given the hidden variable relates to natural images.

For **Flat** patches, the covariance is trivial - it is merely the texture of the stationary texture process $\Sigma$ multiplied by the corresponding contrast scalar $z$. Since we require the texture to be stationary its eigenvectors are the Fourier basis vectors [18] (up to boundary effects), much like the ones visible in the first two components in Figure 5.

For **Edge** patches, given the hidden variable we know which pixel belongs to which "object" in the patch, that is, we know the shape of the occlusion mask exactly. If $i$ and $j$ are two pixels in different objects, we know they will be independent, and as such uncorrelated, resulting in zero entries in the covariance matrix. Thus, if we arrange the pixels by their object assignment, the eigenvectors of such a covariance matrix would be of the form:

$$\begin{bmatrix} \mathbf{0} \\ \mathbf{v} \end{bmatrix} \text{ or } \begin{bmatrix} \mathbf{v} \\ \mathbf{0} \end{bmatrix}$$

where $\mathbf{v}$ is an eigenvector of the stationary (within-object) covariance and the rest of the entries are zeros, thus eigenvectors of the covariance will be zero on one side of the occlusion mask and Fourier-like on the other side. Figure 7c depicts the eigenvector of such an edge component covariance - note the similar structure to Figure 7d and 5. This block structure is a common structure in the GMM learned from natural images, showing that indeed such a dead leaves model is consistent with what we find in GMMs learned on natural images.

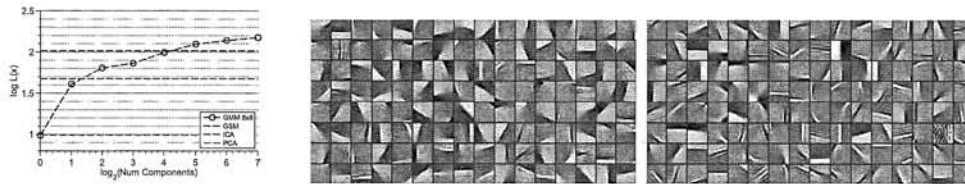

(a) Log Likelihood Comparison      (b) Mini Dead Leaves - ICA      (c) Natural Images - ICA

Figure 8: (a) Log likelihood comparison with mini dead leaves data. We train a GMM with a varying number of components from mini dead leaves samples, and test its likelihood on a test set. We compare to a PCA, ICA and a GSM model, all trained on mini dead leaves samples - as can be seen, the GMM outperforms these considerably. Both PCA and ICA seek *linear* transformations, but since the underlying generative process is *non-linear* (see Figure 7a), they fail. The GSM captures the contrast variation of the data, but does not capture occlusions, which are an important part of this model. (b) and (c) ICA filters learned on mini dead leaves and natural image patches respectively, note the high similarity.

### 4.4  From mini dead leaves to natural images

We repeat the log likelihood experiment from sections 2 and 3, comparing to PCA, ICA and GSM models to GMMs. This time, however, both the training set and test set are generated from the mini dead leaves model. Results can be seen in Figure 8a. Both ICA and PCA do the best job that they can in terms of finding *linear* projections that decorrelate the data (or make it as sparse as possible). But because the *true* generative process for the mini dead leaves is *not* a linear transformation of IID variables, neither of these does a very good job in terms of log likelihood. Interestingly - ICA filters learned on mini dead leaves samples are astonishingly similar to those obtain when trained on natural images - see Figure 8b and 8c. The GSM model can capture the contrast variation of the data easily, but not the structure due to occlusion. A GMM with enough components, on the other hand, is capable of explicitly modeling contrast and occlusion using covariance functions such as in Figure 7c, and thus gives much better log likelihood to the dead leaves data. This exact same pattern of results can be seen in natural image patches (Figure 2), suggesting that the main reason for the excellent performance of GMMs on natural image patches is its ability to model both contrast and occlusions.

## 5  Discussion

In this paper we have provided some additional evidence for the surprising success of GMMs in modeling natural images. We have investigated the causes for this success and the different properties of natural images which are captured by the model. We have also presented an analytical generative model for image patches which explains many of the features learned by the GMM from natural images, as well as the shortcomings of other models.

One may ask - is the mini dead leaves model a good model for natural images? Does it explain everything learned by the GMM? While the mini dead leaves model definitely explains some of the properties learned by the GMM, at its current simple form presented here, it is not a much better model than a simple GSM model. When adding the occlusion process into the model, the mini dead leaves gains ~0.1 bit/pixel when compared to the GSM texture process it uses on its own. This makes it as good as a 32 component GMM, but significantly worse than the 200 components model (for $8 \times 8$ patches). There are two possible explanations for this. One is that the GSM texture process is just not enough, and a richer texture process is needed (much like the one learned by the GMM). The second is that the simple occlusion model we use here is too simplistic, and does not allow for capturing the variable structures of occlusion present in natural images. Both of these may serve as a starting point for a more efficient and explicit model for natural images, handling occlusions and different texture processes explicitly. There have been several works in this direction already [19, 20, 21], and we feel this may hold promise for creating links to higher level visual tasks such as segmentation, recognition and more.

### Acknowledgments

The authors wish to thank the Charitable Gatsby Foundation and the ISF for support.

# References

[1] M. Bethge, "Factorial coding of natural images: how effective are linear models in removing higher-order dependencies?" vol. 23, no. 6, pp. 1253–1268, June 2006.

[2] P. Berkes, R. Turner, and M. Sahani, "On sparsity and overcompleteness in image models," in *NIPS*, 2007.

[3] S. Lyu and E. P. Simoncelli, "Nonlinear extraction of üindependent componentsü of natural images using radial Gaussianization," *Neural Computation*, vol. 21, no. 6, pp. 1485–1519, Jun 2009.

[4] D. Zoran and Y. Weiss, "From learning models of natural image patches to whole image restoration," in *Computer Vision (ICCV), 2011 IEEE International Conference on*. IEEE, 2011, pp. 479–486.

[5] B. Culpepper, J. Sohl-Dickstein, and B. Olshausen, "Building a better probabilistic model of images by factorization," in *Computer Vision (ICCV), 2011 IEEE International Conference on*. IEEE, 2011.

[6] L. Theis, S. Gerwinn, F. Sinz, and M. Bethge, "In all likelihood, deep belief is not enough," *The Journal of Machine Learning Research*, vol. 999888, pp. 3071–3096, 2011.

[7] G. Matheron, *Random sets and integral geometry*. Wiley New York, 1975, vol. 1.

[8] X. Pitkow, "Exact feature probabilities in images with occlusion," *Journal of Vision*, vol. 10, no. 14, 2010.

[9] B. Olshausen *et al.*, "Emergence of simple-cell receptive field properties by learning a sparse code for natural images," *Nature*, vol. 381, no. 6583, pp. 607–609, 1996.

[10] A. J. Bell and T. J. Sejnowski, "The independent components of natural scenes are edge filters," *Vision Research*, vol. 37, pp. 3327–3338, 1997.

[11] A. Hyvarinen and E. Oja, "Independent component analysis: algorithms and applications," *Neural networks*, vol. 13, no. 4-5, pp. 411–430, 2000.

[12] Y. Karklin and M. Lewicki, "Emergence of complex cell properties by learning to generalize in natural scenes," *Nature*, November 2008.

[13] J. Sohl-Dickstein and B. Culpepper, "Hamiltonian annealed importance sampling for partition function estimation," 2011.

[14] M. Lewicki and B. Olshausen, "Probabilistic framework for the adaptation and comparison of image codes," *JOSA A*, vol. 16, no. 7, pp. 1587–1601, 1999.

[15] A. Lee, D. Mumford, and J. Huang, "Occlusion models for natural images: A statistical study of a scale-invariant dead leaves model," *International Journal of Computer Vision*, vol. 41, no. 1, pp. 35–59, 2001.

[16] C. Zetzsche, E. Barth, and B. Wegmann, "The importance of intrinsically two-dimensional image features in biological vision and picture coding," in *Digital images and human vision*. MIT Press, 1993, p. 138.

[17] E. Simoncelli, "Bayesian denoising of visual images in the wavelet domain," *Lecture Notes in Statistics - New York-Springer Verlag*, pp. 291–308, 1999.

[18] D. Field, "What is the goal of sensory coding?" *Neural computation*, vol. 6, no. 4, pp. 559–601, 1994.

[19] J. Lücke, R. Turner, M. Sahani, and M. Henniges, "Occlusive components analysis," *Advances in Neural Information Processing Systems*, vol. 22, pp. 1069–1077, 2009.

[20] G. Puertas, J. Bornschein, and J. Lücke, "The maximal causes of natural scenes are edge filters," in *NIPS*, vol. 23, 2010, pp. 1939–1947.

[21] N. Le Roux, N. Heess, J. Shotton, and J. Winn, "Learning a generative model of images by factoring appearance and shape," *Neural Computation*, vol. 23, no. 3, pp. 593–650, 2011.

